# Learning Shared Latent Structure for Image Synthesis and Robotic Imitation

**Aaron P. Shon** †    **Keith Grochow** †    **Aaron Hertzmann** ‡    **Rajesh P. N. Rao** †

†Department of Computer Science and Engineering
University of Washington
Seattle, WA 98195 USA
‡Department of Computer Science
University of Toronto
Toronto, ON M5S 3G4 Canada
{*aaron,keithg,rao*}*@cs.washington.edu, hertzman@dgp.toronto.edu*

## Abstract

We propose an algorithm that uses Gaussian process regression to learn common hidden structure shared between corresponding sets of heterogenous observations. The observation spaces are linked via a single, reduced-dimensionality latent variable space. We present results from two datasets demonstrating the algorithms's ability to synthesize novel data from learned correspondences. We first show that the method can learn the nonlinear mapping between corresponding views of objects, filling in missing data as needed to synthesize novel views. We then show that the method can learn a mapping between human degrees of freedom and robotic degrees of freedom for a humanoid robot, allowing robotic imitation of human poses from motion capture data.

## 1   Introduction

Finding common structure between two or more concepts lies at the heart of analogical reasoning. Structural commonalities can often be used to interpolate novel data in one space given observations in another space. For example, predicting a 3D object's appearance given corresponding poses of another, related object relies on learning a parameterization common to both objects. Another domain where finding common structure is crucial is imitation learning, also called "learning by watching" [11, 12, 6]. In imitation learning, one agent, such as a robot, learns to perform a task by observing another agent, for example, a human instructor. In this paper, we propose an efficient framework for discovering parameterizations shared between multiple observation spaces using Gaussian processes.

Gaussian processes (GPs) are powerful models for classification and regression that subsume numerous classes of function approximators, such as single hidden-layer neural networks and RBF networks [8, 15, 9]. Recently, Lawrence proposed the Gaussian process latent variable model (GPLVM) [4] as a new technique for nonlinear dimensionality reduction and data visualization [13, 10]. An extension of this model, the scaled GPLVM (SGPLVM), has been used successfully for dimensionality reduction on human motion capture data for motion synthesis and visualization [1].

In this paper, we propose a generalization of the GPLVM model that can handle multiple observation spaces, where each set of observations is parameterized by a different set of kernel parameters. Observations are linked via a single, reduced-dimensionality latent variable space. Our framework can be viewed as a nonlinear extension to canonical correlation

analysis (CCA), a framework for learning correspondences between sets of observations. Our goal is to find correspondences on testing data, given a limited set of corresponding training data from two observation spaces. Such an algorithm can be used in a variety of applications, such as inferring a novel view of an object given a corresponding view of a different object and estimating the kinematic parameters for a humanoid robot given a human pose.

Several properties motivate our use of GPs. First, finding latent representations for correlated, high-dimensional sets of observations requires non-linear mappings, so linear CCA is not viable. Second, GPs reduce the number of free parameters in the regression model, such as number of basis units needed, relative to alternative regression models such as neural networks. Third, the probabilistic nature of GPs facilitates learning from multiple sources with potentially different variances. Fourth, probabilistic models provide an estimate of uncertainty in classification or interpolating between data; this is especially useful in applications such as robotic imitation where estimates of uncertainty can be used to decide whether a robot should attempt a particular pose or not. GPs can also generate samples of novel data, unlike many nonlinear dimensionality reduction methods [10, 13].

Fig. 1(a) shows the graphical model for learning shared structure using Gaussian processes. A latent space $X$ maps to two (or more) observation spaces $Y, Z$ using nonlinear kernels, and "inverse" Gaussian processes map back from observations to latent coordinates. Synthesis employs a map from latent coordinates to observations, while recognition employs an inverse mapping. We demonstrate our approach on two datasets. The first is an image dataset containing corresponding views of two different objects. The challenge is to predict corresponding views of the second object given novel views of the first based on a limited training set of corresponding object views. The second dataset consists of human poses derived from motion capture data and corresponding kinematic poses from a humanoid robot. The challenge is to estimate the kinematic parameters for robot pose, given a potentially novel pose from human motion capture, thereby allowing robotic imitation of human poses. Our results indicate that the model generalizes well when only limited training correspondences are available, and that the model remains robust when testing data is noisy.

## 2   Latent Structure Model

The goal of our model is to find a shared latent variable parameterization in a space $X$ that relates corresponding pairs of observations from two (or more) different spaces $Y, Z$. The observation spaces might be very dissimilar, despite the observations sharing a common structure or parameterization. For example, a robot's joint space may have very different degrees of freedom than a human's joint space, although they may both be made to assume similar poses. The latent variable space then characterizes the common pose space.

Let $\mathbf{Y}, \mathbf{Z}$ be matrices of observations (training data) drawn from spaces of dimensionality $D_Y, D_Z$ respectively. Each row represents one data point. These observations are drawn so that the first observation $\mathbf{y}_1$ corresponds to the observation $\mathbf{z}_1$, observation $\mathbf{y}_2$ corresponds to observation $\mathbf{z}_2$, etc. up to the number of observations $N$. Let $X$ be a "latent space" of dimensionality $D_X \ll D_Y, D_Z$. We initialize a matrix of latent points $\mathbf{X}$ by averaging the top $D_X$ principal components of $\mathbf{Y}, \mathbf{Z}$. As with the original GPLVM, we optimize over a limited subset of training points (the *active set*) to accelerate training, determined by the informative vector machine (IVM) [5]. The SGPLVM assumes that a diagonal "scaling matrix" $\mathbf{W}$ scales the variances of each dimension $k$ of the $\mathbf{Y}$ matrix (a similar matrix $\mathbf{V}$ scales each dimension $m$ of $\mathbf{Z}$). The scaling matrix helps in domains where different output dimensions (such as the degrees of freedom of a robot) can have vastly different variances.

We assume that each latent point $\mathbf{x}_i$ generates a pair of observations $\mathbf{y}_i, \mathbf{z}_i$ via a nonlinear function parameterized by a kernel matrix. GPs parameterize the functions $f_Y : X \mapsto Y$ and $f_Z : X \mapsto Z$. The SGPLVM model uses an exponential (RBF) kernel, defining the

similarity between two data points $\mathbf{x}, \mathbf{x}'$ as:

$$k\left(\mathbf{x}, \mathbf{x}'\right) = \alpha_Y \exp\left(-\frac{\gamma_Y}{2}||\mathbf{x} - \mathbf{x}'||^2\right) + \delta_{\mathbf{x}, \mathbf{x}'}\beta_Y^{-1} \tag{1}$$

given hyperparameters for the $\mathbf{Y}$ space $\theta_Y = \{\alpha_Y, \beta_Y, \gamma_Y\}$. $\delta$ represents the delta function. Following standard notation for GPs [8, 15, 9], the priors $P(\theta_Y), P(\theta_Z), P(\mathbf{X})$, the likelihoods $P(\mathbf{Y}), P(\mathbf{Z})$ for the $\mathbf{Y}, \mathbf{Z}$ observation spaces, and the joint likelihood $P_{GP}(\mathbf{X}, \mathbf{Y}, \mathbf{Z}, \theta_Y, \theta_Z)$ are given by:

$$P(\mathbf{Y}|\theta_Y, \mathbf{X}) = \frac{|\mathbf{W}|^N}{\sqrt{(2\pi)^{ND_Y}|\mathbf{K}|^{D_Y}}} \exp\left(-\frac{1}{2}\sum_{k=1}^{D_Y} w_k^2 \mathbf{Y}_k^{\mathrm{T}}\mathbf{K}_Y^{-1}\mathbf{Y}_k\right) \tag{2}$$

$$P(\mathbf{Z}|\theta_Z, \mathbf{X}) = \frac{|\mathbf{V}|^N}{\sqrt{(2\pi)^{ND_Z}|\mathbf{K}|^{D_Z}}} \exp\left(-\frac{1}{2}\sum_{m=1}^{D_Z} v_m^2 \mathbf{Z}_m^{\mathrm{T}}\mathbf{K}_Z^{-1}\mathbf{Z}_m\right) \tag{3}$$

$$P(\theta_Y) \propto \frac{1}{\alpha_Y\beta_Y\gamma_Y} \qquad P(\theta_Z) \propto \frac{1}{\alpha_Z\beta_Z\gamma_Z} \tag{4}$$

$$P(\mathbf{X}) = \frac{1}{\sqrt{2\pi}}\exp\left(-\frac{1}{2}\sum_i ||\mathbf{x}_i||^2\right) \tag{5}$$

$$P_{GP}(\mathbf{X}, \mathbf{Y}, \mathbf{Z}, \theta_Y, \theta_Z) = P(\mathbf{Y}|\theta_Y, \mathbf{X})P(\mathbf{Z}|\theta_Z, \mathbf{X})P(\theta_Y)P(\theta_Z)P(\mathbf{X}) \tag{6}$$

where $\alpha_Z, \beta_Z, \gamma_Z$ are hyperparameters for the $Z$ space, and $w_k, v_m$ respectively denote the diagonal entries for matrices $\mathbf{W}, \mathbf{V}$. Let $\overline{\mathbf{Y}}, \overline{\mathbf{K}}_Y^{-1}$ respectively denote the $\mathbf{Y}$ observations from the active set (with mean $\mu_Y$ subtracted out) and the kernel matrix for the active set. The joint negative log likelihood of a latent point $\mathbf{x}$ and observations $\mathbf{y}, \mathbf{z}$ is:

$$L_{\mathbf{y}|\mathbf{x}}(\mathbf{x}, \mathbf{y}) = \frac{||\mathbf{W}(\mathbf{y} - f_Y(\mathbf{x}))||^2}{2\sigma_Y^2(\mathbf{x})} + \frac{D_Y}{2}\ln\left(\sigma_Y^2(\mathbf{x})\right) \tag{7}$$

$$f_Y(\mathbf{x}) = \mu_Y + \overline{\mathbf{Y}}^{\mathrm{T}}\overline{\mathbf{K}}_Y^{-1}\mathbf{k}(\mathbf{x}) \tag{8}$$

$$\sigma_Y^2(\mathbf{x}) = k(\mathbf{x}, \mathbf{x}) - \mathbf{k}(\mathbf{x})^{\mathrm{T}}\overline{\mathbf{K}}_Y^{-1}\mathbf{k}(\mathbf{x}) \tag{9}$$

$$L_{\mathbf{z}|\mathbf{x}}(\mathbf{x}, \mathbf{z}) = \frac{||\mathbf{V}(\mathbf{z} - f_Z(\mathbf{x}))||^2}{2\sigma_Z^2(\mathbf{x})} + \frac{D_Z}{2}\ln\left(\sigma_Z^2(\mathbf{x})\right) \tag{10}$$

$$f_Z(\mathbf{x}) = \mu_Z + \overline{\mathbf{Z}}^{\mathrm{T}}\overline{\mathbf{K}}_Z^{-1}\mathbf{k}(\mathbf{x}) \tag{11}$$

$$\sigma_Z^2(\mathbf{x}) = k(\mathbf{x}, \mathbf{x}) - \mathbf{k}(\mathbf{x})^{\mathrm{T}}\overline{\mathbf{K}}_Z^{-1}\mathbf{k}(\mathbf{x}) \tag{12}$$

$$L_{\mathbf{x}, \mathbf{y}, \mathbf{z}} = L_{\mathbf{y}|\mathbf{x}} + L_{\mathbf{z}|\mathbf{x}} + \frac{1}{2}||\mathbf{x}||^2 \tag{13}$$

The model learns a separate kernel for each observation space, but a single set of common latent points. A conjugate gradient solver adjusts model parameters and latent coordinates to maximize Eq. 6.

Given a trained SGPLVM, we would like to infer the parameters in one observation space given parameters in the other (e.g., infer robot pose $\mathbf{z}$ given human pose $\mathbf{y}$). We solve this problem in two steps. First, we determine the most likely latent coordinate $\mathbf{x}$ given the observation $\mathbf{y}$ using $\mathrm{argmax}_{\mathbf{x}} L_X(\mathbf{x}, \mathbf{y})$. In principle, one could find $\mathbf{x}$ at $\frac{\partial L_X}{\partial \mathbf{x}} = 0$ using gradient descent. However, to speed up recognition, we instead learn a separate "inverse" Gaussian process $f_Y^{-1} : \mathbf{y} \mapsto \mathbf{x}$ that maps back from the space $Y$ to the space $X$. Once the correct latent coordinate $\mathbf{x}$ has been inferred for a given $\mathbf{y}$, the model uses the trained SGPLVM to predict the corresponding observation $\mathbf{z}$.

# 3 Results

We first demonstrate how the our model can be used to synthesize new views of an object, character or scene from known views of another object, character or scene, given a common latent variable model. For ease of visualization, we used 2D latent spaces for all results shown here. The model was applied to image pairs depicting corresponding views of 3D objects. Different views show the objects[1] rotated at varying degrees out of the camera plane. We downsampled the images to $32 \times 32$ grayscale pixels. For fitting images, the scaling matrices $\mathbf{W}, \mathbf{V}$ are of minimal importance (since we expect all pixels should *a priori* have the same variance). We also found empirically that using $f_Y(\mathbf{x}) = \mathbf{Y}^{\mathrm{T}} \overline{\mathbf{K}}_Y^{-1} \mathbf{k}(\mathbf{x})$ instead of Eqn. 8 produced better renderings. We rescaled each $f_Y$ to use the full range of pixel values $[0 \dots 255]$, creating the images shown in the figures.

Fig. 1(b) shows how the model extrapolates to novel datasets given a limited set of training correspondences. We trained the model using 72 corresponding views of two different objects, a coffee cup and a toy truck. Fixing the latent coordinates learned during training, we then selected 8 views of a third object (a toy car). We selected latent points corresponding to those views, and learned kernel parameters for the 8 images. Empirically, priors on kernel parameters are critical for acceptable performance, particularly when only limited data are available such as the 8 different poses for the toy car. In this case, we used the kernel parameters learned for the cup and toy truck (based on 72 different poses) to impose a Gaussian prior on the kernel parameters for the car (replacing $P(\theta)$ in Eqn. 4):

$$-\log P(\theta_{\mathrm{car}}) = -\log P_{GP} + (\theta_{\mathrm{car}} - \theta_\mu)^{\mathrm{T}} \Gamma_\theta^{-1} (\theta_{\mathrm{car}} - \theta_\mu) \qquad (14)$$

where $\theta_{\mathrm{car}}, \theta_\mu, \Gamma_\theta^{-1}$ are respectively kernel parameters for the car, the mean kernel parameters for previously learned kernels (for the cup and truck), and inverse covariance matrix for learned kernel parameters. $\theta_\mu, \Gamma_\theta^{-1}$ in this case are derived from only two samples, but nonetheless successfully constrain the kernel parameters for the car so the model functions on the limited set of 8 example poses.

To test the model's robustness to noise and missing data, we randomly selected 10 latent coordinates corresponding to a subset of learned cup and truck image pairs. We then added varying displacements to the latent coordinates and synthesized the corresponding *novel* views for all 3 observation spaces. Displacements varied from 0 to 0.45 (all 72 latent coordinates lie on the interval [-0.70,-0.87] to [0.72,0.56]). The synthesized views are shown in Fig. 1(b), with images for the cup and truck in the first two rows. Latent coordinates in regions of low model likelihood generate images that appear blurry or noisy. More interestingly, despite the small number of images used for the car, the model correctly matches the orientation of the car to the synthesized images of the cup and truck. Thus, the model can synthesize reasonable correspondences (given a latent point) even if the number of training examples used to learn kernel parameters is small.

Fig. 2 illustrates the recognition performance of the "inverse" Gaussian process model as a function of the amount of noise added to the inputs. Using the latent space and kernel parameters learned for Fig. 1, we present 72 views of the coffee cup with varying amounts of additive, zero-mean white noise, and determine the fraction of the 72 poses correctly classified by the model. The model estimates the pose using 1-nearest-neighbor classification of the latent coordinates $\mathbf{x}$ learned during training:

$$\underset{\mathbf{x}'}{\mathrm{argmax}}\, k\left(\mathbf{x}, \mathbf{x}'\right) \qquad (15)$$

The recognition performance degrades gracefully with increasing noise power. Fig. 2 also plots sample images from one pose of the cup at several different noise levels. For two of the noise levels, we show the "denoised" cup image selected using the nearest-neighbor

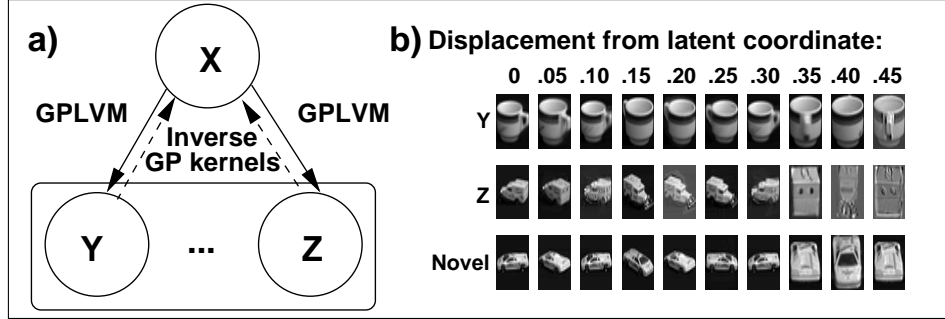

Figure 1: **Pose synthesis for multiple objects using shared structure:** (a) Graphical model for our shared structure latent variable model. The latent space $X$ maps to two (or more) observation spaces $Y, Z$ using a nonlinear kernel. "Inverse" Gaussian process kernels map back from observations to latent coordinates. (b) The model learns pose correspondences for images of the coffee cup and toy truck (**Y** and **Z**) by fitting kernel parameters and a 2-dimensional latent variable space. After learning the latent coordinates for the cup and truck, we fit kernel parameters for a novel object (the toy car). Unlike the cup and truck, where 72 pairs of views were used to fit kernel parameters and latent coordinates, only 8 views were used to fit kernel parameters for the car. The model is robust to noise in the latent coordinates; numbers above each column represent the amount of noise added to the latent coordinates used to synthesize the images. Even at points where the model is uncertain (indicated by the rightmost results in the **Y** and **Z** rows), the learned kernel extrapolates the correct view of the toy car (the "novel" row).

classification, and the corresponding reconstructed truck. This illustrates how even noisy observations in one space can predict corresponding observations in the companion space.

Fig. 3 illustrates the ability of the model to synthesize novel views of one object given a novel view of a different object. A limited set of corresponding poses (24 of 72 total) of a cat figurine and a mug were used to train the GP model. The remaining 48 poses of the mug were then used as testing data. For each snapshot of the mug, we inferred a latent point using the "inverse" Gaussian process model and used the learned model to synthesize what the cat figurine should look like in the same pose. A subset of these results is presented in the rows on the left in Fig. 3: the "Test" rows show novel images of the mug, the "Inferred" rows show the model's best estimate for the cat figurine, and the "Actual" rows show the ground truth. Although the images for some poses are blurry and the model fails to synthesize the correct image for pose 44, the model nevertheless manages to capture fine detail on most of the images.

The grayscale plot at upper right in Fig. 3 shows model certainty $1/\left[\sigma_Y^2(\mathbf{x}) + \sigma_Z^2(\mathbf{x})\right]$, with white where the model is highly certain and black where the model is highly uncertain. Arrows indicate the path in latent space formed by the training images. The dashed line indicates latent points inferred from testing images of the mug. Numbered latent coordinates correspond to the synthesized images at left. The latent space shows structure: latent points for similar poses are grouped together, and tend to move along a smooth curve in latent space, with coordinates for the final pose lying close to coordinates for the first pose (as desired for a cyclic image sequence). The bar graph at lower right compares model certainty for the numbered latent coordinates; higher bars indicate greater model certainty. The model appears particularly uncertain for blurry inferred images, such as 8, 14, and 26.

Fig. 4 shows an application of our framework to the problem of robotic imitation of human actions. We trained our model on a dataset containing human poses (acquired with a Vicon motion capture system) and corresponding poses of a Fujitsu HOAP-2 humanoid robot. Note that the robot has 25 degrees-of-freedom which differ significantly from the degrees-

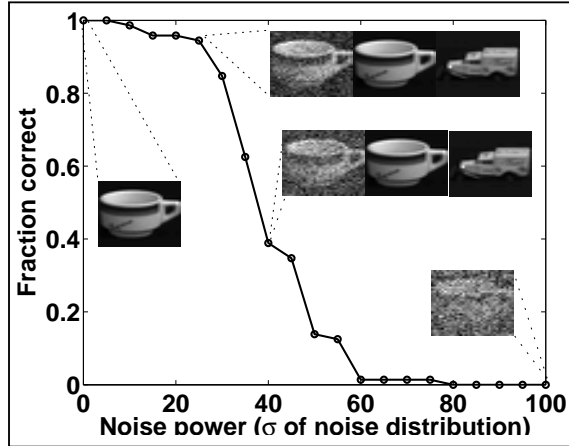

Figure 2: **Recognition using a Learned Latent Variable Space:** After learning from 72 paired correspondences between poses of a coffee cup and of a toy truck, the model is able to recognize different poses of the coffee cup in the presence of additive white noise. Fraction of images recognized are plotted on the Y axis and standard deviation of white noise is plotted on the X axis. One pose of the cup (of 72 total) is plotted for various noise levels (see text for details). "Denoised" images obtained from nearest-neighbor classification and the corresponding images for the $Z$ space (the toy truck) are also shown.

of-freedom of the human skeleton used in motion capture. After training on 43 roughly matching poses (only linear time scaling applied to align training poses), we tested the model by presenting a set of 123 human motion capture poses (which includes the original training set). Because the recognition model $f_Y^{-1} : \mathbf{y} \mapsto \mathbf{x}$ is not trained from samples from the prior distribution of the data, $P(\mathbf{x}, \mathbf{y})$, we found it necessary to approximate $\mathbf{k}(\mathbf{x})$ for the recognition model by rescaling $\mathbf{k}(\mathbf{x})$ for the testing points to lie on the same interval as the $\mathbf{k}(\mathbf{x})$ values of the training points. We suspect that providing proper samples from the prior will improve recognition performance. As illustrated in Fig. 4 (inset panels, human and robot skeletons), the model was able to correctly infer appropriate robot kinematic parameters given a range of novel human poses. These inferred parameters were used in conjunction with a simple controller to instantiate the pose in the humanoid robot (see photos in the inset panels).

## 4 Discussion

Our Gaussian process model provides a novel method for learning nonlinear relationships between corresponding sets of data. Our results demonstrate the model's utility for diverse tasks such as image synthesis and robotic programming by demonstration. The GP model is closely related to other kernel methods for solving CCA [3] and similar problems [2].

The problems addressed by our model can also be framed as a type of nonlinear CCA. Our method differs from the latent variable method proposed in [14] by using Gaussian process regression. Disadvantages of our method with respect to [14] include lack of global optimality for the latent embedding; advantages include fewer independent parameters and the ability to easily impose priors on the latent variable space (since GPLVM regression uses conjugate gradient optimization instead of eigendecomposition). Empirically we found the flexiblity of the GPLVM approach desirable for modeling a diversity of data sources.

Our framework learns mappings between each observation space and a latent space, rather than mapping directly between the observation spaces. This makes visualization and interaction much easier. An intermediate mapping to a latent space is also more economical in

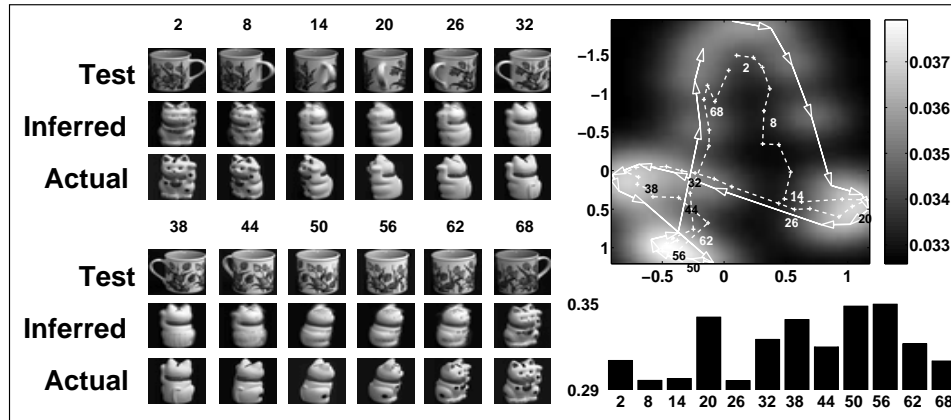

Figure 3: **Synthesis of novel views using a shared latent variable model:** After training on 24 paired images of a mug with a cat figurine (out of 72 total paired images), we ask the model to infer what the remaining 48 poses of the cat would look like given 48 novel views of the mug. The system uses an inverse Gaussian process model to infer a 2D latent point for each of the 48 novel mug views, then synthesizes a corresponding view of the cat figurine. At left we plot the novel testing mug images given to the system ("test"), the synthesized cat images ("inferred"), and the actual views of the cat figurine from the database ("actual"). At upper right we plot the model uncertainty in the latent space. The 24 latent coordinates from the training data are plotted as arrows, while the 48 novel latent points are plotted as crosses on a dashed line. At lower right we show model certainty for the cat figurine data $(1/\sigma_Z^2(\mathbf{x}))$ for each testing latent point $\mathbf{x}$. Note the low certainty for the blurry inferred images labeled 8, 14, and 26.

the limit of many correlated observation spaces. Rather than learning all pairwise relations between observation spaces (requiring a number of parameters quadratic in the number of observation spaces), our method learns one generative and one inverse mapping between each observation space and the latent space (so the number of parameters grows linearly).

From a cognitive science perspective, such an approach is similar to the Active Intermodal Mapping (AIM) hypothesis of imitation [6]. In AIM, an imitating agent maps its own actions and its perceptions of others' actions into a single, modality-independent space. This modality-independent space is analogous to the latent variable space in our model. Our model does not directly address the "correspondence problem" in imitation [7], where correspondences between an agent and a teacher are established through some form of unsupervised feature matching. However, it is reasonable to assume that imitation by a robot of human activity could involve some initial, explicit correspondence matching based on simultaneity. Turn-taking behavior is an integral part of human-human interaction. Thus, to bootstrap its database of corresponding data points, a robot could invite a human to take turns playing out motor sequences. Initially, the human would imitate the robot's actions and the robot could use this data to learn correspondences using our GP model; later, the robot could check and if necessary, refine its learned model by attempting to imitate the human's actions.

**Acknowledgements:** This work was supported by NSF AICS grant no. 130705 and an ONR YIP award/NSF Career award to RPNR. We thank the anonymous reviewers for their comments.

**References**

[1] K. Grochow, S. L. Martin, A. Hertzmann, and Z. Popović. Style-based inverse kinematics. In *Proc. SIGGRAPH*, 2004.

[2] J. Ham, D. Lee, and L. Saul. Semisupervised alignment of manifolds. In *AISTATS*, 2004.

[3] P. L. Lai and C. Fyfe. Kernel and nonlinear canonical correlation analysis. *Int. J. Neural Sys.*, 10(5):365–377, 2000.

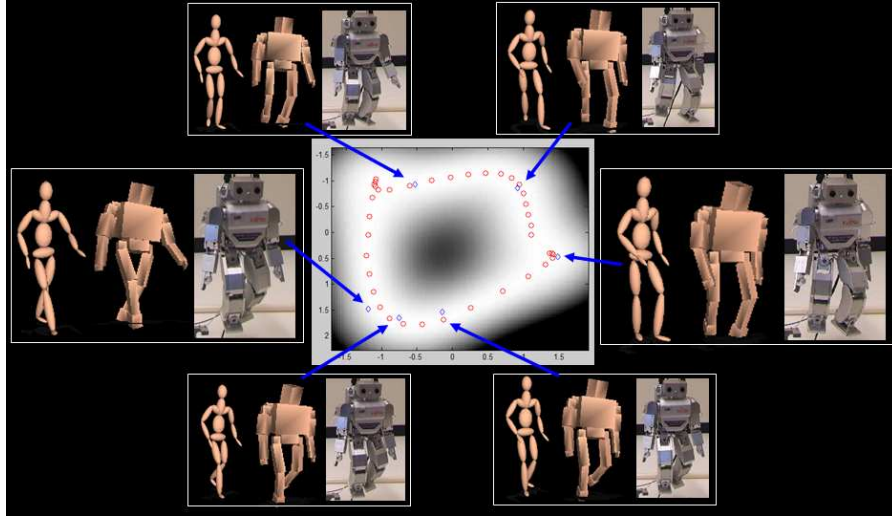

Figure 4: **Learning shared latent structure for robotic imitation of human actions:** The plot in the center shows the latent training points (red circles) and model precision $1/\sigma_Z^2$ for the robot model (grayscale plot), with examples of recovered latent points for testing data (blue diamonds). Model precision is qualitatively similar for the human model. Inset panels show the pose of the human motion capture skeleton, the simulated robot skeleton, and the humanoid robot for each example latent point. The model correctly infers robot poses from the human walking data (inset panels).

[4] N. D. Lawrence. Gaussian process models for visualization of high dimensional data. In S. Thrun, L. Saul, and B. Schölkopf, editors, *Advances in NIPS 16*.

[5] N. D. Lawrence, M. Seeger, and R. Herbrich. Fast sparse Gaussian process methods: the informative vector machine. In S. Becker, S. Thrun, and K. Obermayer, editors, *Advances in NIPS 15*, 2003.

[6] A. N. Meltzoff. Elements of a developmental theory of imitation. In A. N. Meltzoff and W. Prinz, editors, *The imitative mind: Development, evolution, and brain bases*, pages 19–41. Cambridge: Cambridge University Press, 2002.

[7] C. Nehaniv and K. Dautenhahn. The correspondence problem. In *Imitation in Animals and Artifacts*. MIT Press, 2002.

[8] A. O'Hagan. On curve fitting and optimal design for regression. *Journal of the Royal Statistical Society B*, 40:1–42, 1978.

[9] C. E. Rasmussen. *Evaluation of Gaussian Processes and other Methods for Non-Linear Regression*. PhD thesis, University of Toronto, 1996.

[10] S. Roweis and L. Saul. Nonlinear dimensionality reduction by locally linear embedding. *Science*, 290(5500):2323–2326, 2000.

[11] S. Schaal, A. Ijspeert, and A. Billard. Computational approaches to motor learning by imitation. *Phil. Trans. Royal Soc. London: Series B*, 358:537–547, 2003.

[12] A. P. Shon, D. B. Grimes, C. L. Baker, and R. P. N. Rao. A probabilistic framework for model-based imitation learning. In *Proc. 26th Ann. Mtg. Cog. Sci. Soc.*, 2004.

[13] J. B. Tenenbaum, V. de Silva, and J. C. Langford. A global geometric framework for nonlinear dimensionality reduction. *Science*, 290(5500):2319–2323, 2000.

[14] J. J. Verbeek, S. T. Roweis, and N. Vlassis. Non-linear CCA and PCA by alignment of local models. In *Advances in NIPS 16*, pages 297–304. 2003.

[15] C. K. I. Williams. Computing with infinite networks. In M. C. Mozer, M. I. Jordan, and T. Petsche, editors, *Advances in NIPS 9*. Cambridge, MA: MIT Press, 1996.

## Footnotes

[1]http://www1.cs.columbia.edu/CAVE/research/softlib/coil-100.html
